# Surface Reconstruction using Learned Shape Models

**Jan Erik Solem**
School of Technology and Society
Malmö University, Sweden
jes@ts.mah.se

**Fredrik Kahl**
RSISE, Australian National University
ACT 0200, Australia
fredrik@maths.lth.se

## Abstract

We consider the problem of geometrical surface reconstruction from one or several images using learned shape models. While humans can effortlessly retrieve 3D shape information, this inverse problem has turned out to be difficult to perform automatically. We introduce a framework based on level set surface reconstruction and shape models for achieving this goal. Through this merging, we obtain an efficient and robust method for reconstructing surfaces of an object category of interest.

The shape model includes surface cues such as point, curve and silhouette features. Based on ideas from Active Shape Models, we show how both the geometry and the appearance of these features can be modelled consistently in a multi-view context. The complete surface is obtained by evolving a level set driven by a PDE, which tries to fit the surface to the inferred 3D features. In addition, an a priori 3D surface model is used to regularize the solution, in particular, where surface features are sparse. Experiments are demonstrated on a database of real face images.

## 1 Introduction

Humans have a remarkable ability of perceiving 3D shape information - even from a single photograph. Exactly how the human visual system works or how shape is represented is to a large extent unknown. It is clear that the capabilities rely on a strong prior model and the efficient use of different surface cues. The corresponding task of automatically recovering a 3D surface model for a computer has turned out to be a challenging problem, even with the addition of multiple images. In the present paper, we try to solve the problem in the case of a single object category, more specifically, faces. At the heart of our approach lies the combination of sophisticated surface reconstruction techniques and a strong statistical model of surface features.

The first part concerns the statistical model of surface features for inferring 3D shape. The features are primarily geometric ones, such as point, curve and silhouette features. Both the geometric relations and their appearances (in terms of image intensities) are modelled. The distributions are learned from real data. Also, a 3D model of the complete surface is used as a weak regularizer where surface features are sparse. The motivations for introducing such a model are several. We are interested in automatically recovering a surface model given new image data of the object of interest. It is a hard problem to robustly extract curves and apparent contours (i.e. silhouettes) without any a priori model. Moreover, many objects are

hard to reliably reconstruct due to specularities and illumination effects. By using distinct geometric features and strong priors, we will still be able to obtain reliable results. Another problem is textureless areas, and more generally, lack of information in the input. Our a priori model will work as a domain-specific regularizer.

The second part of this work deals with fitting surfaces to points and curves in 3D space, and at the same time, fitting the projections of surface contours to apparent contours in the images. The approach taken here is a variational one - we define a functional which we will try to minimize with respect to some parameters describing the geometry of the surfaces. This variational problem leads to a surface evolution, driven by a PDE. The surface is represented implicitly as the level set of a real-valued function [1].

### 1.1 Related Work

In the area of statistical shape models, our work is related to and inspired by Active Shape Models (ASM) [2]. One distinction is that we will model both 2D and 3D data, while ASM have mainly have been applied to 2D objects. In contrast to standard ASM, our observations are from multiple views. In [3] a multi-view model is utilized, but no explicit (or even consistent) 3D data is maintained within the model. In order to do inference on our model, we have adopted the ideas of Probabilistic PCA (PPCA) [4].

In the seminal work [5], a complete 3D model is built from a database of 200 laser scans of faces. The so-called morphable model can be fitted to an image with very impressive results. The model itself is quite complex (approximately 70000 vertices), resulting in long computation times. Although recent advances, their method still requires manual intervention [6]. A generic face model has also been used in [7] for computing regularized structure and motion as well as in [8] based on silhouettes obtained from background subtraction.

In the area of using level set surface representations for fitting surfaces to 3D data, this paper is related to the work in [9] where surfaces are fitted to points using a distance potential. In [10] apparent contours were incorporated for level set based surface fitting. Surfaces can also be estimated using 2D dense data, as in [11] based on photo-consistency. Shape priors for level sets have previously been applied to segmentation, e.g. [12].

### 1.2 Contribution of the Paper

The main contribution of this paper is the approach itself - the combination of a state-of-the-art surface reconstruction technique and the learned statistical model of surface features. This merging results in a robust and efficient method for surface reconstruction without any need for manual intervention. There is no need for an abundant number of images, in fact, a single image is sufficient.

Another key contribution is the introduction of a multi-view feature model which is capable of representing both 2D and 3D data in a consistent manner. As the model is fully probabilistic, the missing data problem can be handled in a natural way. By only incorporating distinct surface features, compared to a full morphable model, we not only gain computational efficiency, but also robustness to specularities and other illumination effects. This point is a also valid when compared to other surface reconstruction methods based on image-correlation [11]. In the case of face modelling, this is known to cause problems due to the complexity of the BRDF for human skin. The main contribution within the field of level sets is in the incorporation of an a priori 3D model used for surface regularization.

## 2 Part I: A Learned Shape Model

In this section, we develop a statistical model which is needed in order to be able to automatically extract and compute 3D surface features from the input images. The output

from the model will serve as input to the level set reconstruction algorithm in order to get a complete surface model. In addition, we will use an a priori 3D surface model as a (weak) regularizer.

## 2.1 The Feature Model

Suppose we have a number of elements in a $d$-dimensional vector $\mathbf{t}$, for example, a collection of 3D point coordinates. Suppose $\mathbf{t}$ can be related to some latent vector $\mathbf{u}$ of dimension $q$ where the relationship is linear:

$$\mathbf{t} = W\mathbf{u} + \mu, \tag{1}$$

where $W$ is a matrix of size $d \times q$ and $\mu$ is $d$-vector allowing for non-zero mean. However, our measurements take place in the images, which is a non-linear function of the 3D features according to the perspective camera model. Denote the projection function with $f : \mathbf{R}^d \to \mathbf{R}^e$, projecting all 3D features to 2D image features, for one or several images[1]. Also, we need to change coordinate system of the 3D features to suit the actual projection function. Denote this mapping by $T : \mathbf{R}^d \to \mathbf{R}^d$. Thus, $f(T(\mathbf{t}))$ will project all normalized 3D data to all images.

Finally, a noise model needs to be specified. We assume that the image measurements are independent and normally distributed, likewise, the latent variables are assumed to be Gaussian with unit variance $\mathbf{u} \sim N(0, I)$. Thus, in summary:

$$\mathbf{t_{2D}} = f(T(\mathbf{t})) + \epsilon = f(T(W\mathbf{u} + \mu)) + \epsilon, \tag{2}$$

where $\epsilon \sim N(0, \sigma^2 I)$ for some scalar $\sigma$.

Before the model can be used, its parameters need to be estimated from training data. Given that it is a probabilistic model, this is best done with maximum likelihood (ML). Suppose we are given $n$ examples $\{\mathbf{t_{2D},i}\}_{i=1}^n$, the ML estimate for $W$ and $\mu$ is obtained by minimizing:

$$\sum_{i=1}^n \left( \frac{1}{\sigma^2} ||(\mathbf{t_{2D},i} - f(T_i(W\mathbf{u}_i + \mu))||^2 + ||\mathbf{u}_i||^2 \right), \tag{3}$$

over all unknowns. The standard deviation $\sigma$ is estimated a priori from the data. Once the model parameters $W$ and $\mu$ have been learned from examples, they are kept fix. In practice, to minimize (3) we alternatively optimize over $(W, \mu)$ and the latent variables $\{\mathbf{u}_i\}_i^n$ using gradient descent. Initial estimates can be obtained by intersecting 3D structure from each set of images and then applying standard PPCA algorithms for the linear part [4]. The normalization $T_i(\cdot)$ is chosen such that each normalized 3D sample has zero mean and unit variance.

A 3D point which is visible in $m > 1$ images will be represented in the vector $\mathbf{t}$ with its 3D coordinates $(X, Y, Z)$. For points visible in only one image, $m = 1$, no depth information is available, and such points are represented similarly to apparent contour points. A curve will be represented in the model by a number of points along the curve. In the training of the model, it is important to parameterize each 3D curve such that each point on the curve approximately corresponds to the same point on the corresponding curve in the other examples. As for curves, we sample the apparent contours (in the images) using arc-length parametrization. However, there is no 3D information available for the apparent contours as they are view-dependent. A simple way is to treat contours points as 3D points with a constant, approximate (but crude) depth estimate.

## 2.2 The Grey-Level Model

The missing component in the model is the relationship between 2D image features and the underlying grey-level (or color) values at these pixels. Again, we adopt a linear model (PPCA). Using the same notation as in (1), but now with the subscript *gl* for grey-level, the model can be written

$$\mathbf{t}_{gl} = W_{gl}\mathbf{u}_{gl} + \mu_{gl} + \epsilon_{gl} \ ,$$

where $\mathbf{t}_{gl}$ is a vector containing the grey-level values of all the 2D image features and $\epsilon_{gl}$ is Gaussian noise in the measurements. In the training phase, each data sample of grey-levels is normalized by subtracting the mean and scaling to unit variance. The ML-estimate of $W_{gl}$ and $\mu_{gl}$ is computed with the EM-algorithm [4]. The complete statistical two-layer model with one feature model and one grey-level model is very similar to the concept of ASM [2]. In principle, the same techniques as used for ASM can be applied to automatically compute the latent variables of the system, i.e. $\mathbf{u}$ and $\mathbf{u}_{gl}$.

## 2.3 The 3D Model

The two-layer feature model produces only a sparse set of features in 3D space. Even if these cues are characteristic for a particular sample (or individual), it is often not enough in order to infer a complete surface model, in particular, in regions where the features are sparse. Therefore, we introduce a 3D surface model consisting of the complete mean surface serving as domain-specific regularizer. The mean surface is obtained from laser scans with the technique described in [13].

# 3 Part II: Surface Reconstruction

## 3.1 Level Set Formulation

Let $\mathbf{x}$ be a point in the open set $\Omega \subset \mathbf{R}^3$. The time dependent surface $\Gamma(t)$ is represented implicitly as the zero level set of a function $\phi(\mathbf{x}, t) : \Omega \times \mathbf{R}_+ \to \mathbf{R}$ as

$$\Gamma(t) = \{\mathbf{x} \ ; \ \phi(\mathbf{x}, t) = 0\} \ , \tag{4}$$

where $\phi$ is defined such that $\phi(\mathbf{x}, t) < 0$ inside $\Gamma$ and $\phi(\mathbf{x}, t) > 0$ outside. Using the definition above gives the outward unit normal $\mathbf{n}$ and the mean curvature $\kappa$ as

$$\mathbf{n} = \frac{\nabla\phi}{|\nabla\phi|} \quad \text{and} \quad \kappa = \nabla \cdot \frac{\nabla\phi}{|\nabla\phi|} \ . \tag{5}$$

One important, frequently used example is the signed distance function, where the requirement $|\nabla\phi(\mathbf{x})| = 1$ is imposed.

The zero set of $\phi(\mathbf{x}, t)$ represents $\Gamma(t)$ at all times $t$. This means that $\phi(\mathbf{x}(t), t) \equiv 0$ for a point on the curve $\mathbf{x}(t) \in \Gamma(t)$. Differentiating with respect to $t$ gives

$$\phi_t + \mathbf{v} \cdot \nabla\phi = 0 \ \Leftrightarrow \ \phi_t + v_n|\nabla\phi| = 0 \ , \tag{6}$$

where $\mathbf{v} = d\mathbf{x}(t)/dt$ and $v_n$ is the velocity normal to the surface. This PDE is solved in order to move the surface according to some derived velocity $\mathbf{v}$. For a more thorough treatment of level set surfaces, see [1].

## 3.2 Surface Fitting to Points

In [9] surfaces are fitted to a point set $\mathcal{S}$ using the level set method. An initial surface is deformed in the gradient direction of an energy functional which involves elastic energy

and potential energy. The energy is expressed using a distance potential as the surface integral

$$E_P(\Gamma) = \int_\Gamma d(\mathbf{x}) \, d\sigma \ , \tag{7}$$

where $\Gamma$ is the surface, $d\sigma$ surface area and $d(\mathbf{x}) = \mathrm{dist}(\mathbf{x}, \mathcal{S})$ is the distance from the point $\mathbf{x}$ to $\mathcal{S}$. The gradient descent of (7) is, cf. [9],

$$\phi_t = (\nabla d(\mathbf{x}) \cdot \mathbf{n} + d(\mathbf{x})\kappa) \, |\nabla \phi| \ , \tag{8}$$

where $\mathbf{n}$ is the surface normal and $\kappa$ the mean curvature.

This motion is known to interfere with surface regularization, since *all* surface points are attracted to the 3D features. Therefore we cut the influence of the potential by setting $d(\mathbf{x}) = \min(d(\mathbf{x}), d_{max})$.

### 3.3  Surface Fitting to Apparent Contours

Let $\gamma$ be an apparent contour in an image parameterized as $\gamma(s) : \ I \subset \mathbf{R} \to \mathbf{R}^2$. The back-projected cone, written in homogeneous coordinates,

$$C(s) = \mathbf{c} + \lambda P^+ \gamma(s) \ , \tag{9}$$

(where $\mathbf{c}$ is the camera center and $P^+$ the pseudo-inverse of the $3 \times 4$ camera matrix of a perspective projection) should graze the surface at the location of the contour generator. It is undesirable to attract the surface to the entire back-projected cone of the apparent contour. The cone should only touch the surface along a curve - the so called contour generator.

We propose to solve this in the following manner. For each point on the curve $\mathbf{m} = \gamma(s)$, let $\mathbf{x}^*$ denote the point closest to $\Gamma$ (If there are several, then choose the one with smallest depth). The function $\phi$ is kept to be a signed distance function, i.e. $|\nabla \phi| = 1$. This means that the point $\mathbf{x}^*$ is easily found by checking the values of $\phi$ along the line of sight given by (9). The set of these points, $\mathcal{S}'$, will be a (possibly discontinuous) space curve $\gamma^*(\lambda, s)$. This set is then added to the distance potential as

$$d(\mathbf{x}) = \min(d(\mathbf{x}), d'(\mathbf{x})) \ , \tag{10}$$

where $d'(\mathbf{x}) = \mathrm{dist}(\mathbf{x}, \mathcal{S}')$ is updated at appropriate intervals as the surface evolves.

### 3.4  Adding a 3D Shape Prior

Since the data, i.e. the points, curves and contours, are sparse it is customary to use a prior for the regions where there is no information. Instead of the common choice of minimal surface type models, we propose to use a learned shape model, as described in Section 2.3. By first aligning the mean shape to the data, the deviation can be expressed similar to (7) as

$$E_{Prior}(\Gamma) = \int_\Gamma d_{Prior}(\mathbf{x}) \, d\sigma \ , \tag{11}$$

where $d_{Prior}(\mathbf{x})$ is the distance potential of the aligned mean shape.

### 3.5  The Combined Motion PDE

Adding all the components above gives a functional $E_{Tot} = E_P + \alpha E_{Prior}$, where $\alpha \in \mathbf{R}_+$ determines the weight of the prior shape. Combining the components (8), (10) and (11) above leads to a PDE for the motion of the surface as

$$\phi_t = [(\nabla d(\mathbf{x}) + \alpha \nabla d_{Prior}(\mathbf{x})) \cdot \mathbf{n} + (d(\mathbf{x}) + \alpha d_{Prior}(\mathbf{x}))\kappa]|\nabla \phi| \ . \tag{12}$$

This PDE is solved iteratively until a steady state is reached, which yields a (local) minimum of $E_{Tot}$.

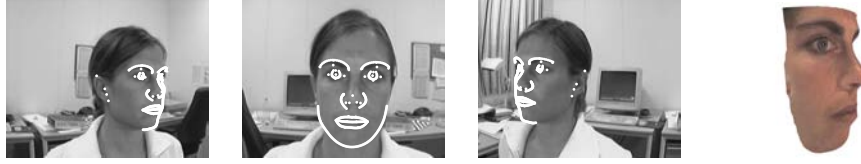

Figure 1: Extracted image features for one image triple and the reconstructed 3D surface.

## 4 Experiments

### 4.1 The Shape Model

All images were taken by a stereo setup with three (synchronized) cameras. The setup was pre-calibrated for both extrinsic and intrinsic camera parameters. Example images are given in Figures 1 and 2. In total a database of 28 image triplets were collected of faces from different persons (23 males and 5 females of ages ranging from 7 to 65). 25 of these were used for training and 3 for testing the two-layer feature model. The 3D mean surface was computed from a database of 24 persons (different from the 28 persons above) using a laser scanner as described in [13].

The 28 triplets in the training and test set were manually labelled - 36 points, 8 curves and 5 apparent contours were extracted for each person, see Figure 1. The two-layer model has $q = 12$ elements in the latent variable $\mathbf{u}$ for the geometrical part and $q_{gl} = 15$ elements in $\mathbf{u}_{gl}$ for the grey-level model. These numbers were found empirically in order to capture most of the variations in the data. The model is able to adopt quite well to the test set. In fact, even for one (frontal) input image of the test set, the model predicts the two profile views remarkably well. As the pose over all faces is (essentially) constant for both test and training images, the normalizing coordinate mapping $T$ is restricted to scale and translation.

### 4.2 Surface Reconstruction

Once 3D data has been obtained by fitting the two-layer shape model to the input images, surfaces are fitted to this data by solving the PDE (12). In the standard level set formulation, the surface must be closed in the computational domain in order to make $\phi$ continuous. We use the technique in [14] for initializing and evolving open implicit surfaces.

Surfaces were reconstructed for a number of persons, a selection is shown in Figures 1 and 2, where the zero set is visualized and triangulated using the marching cubes algorithm . The value of $\alpha$ was 0.3 and $d_{max}$ was set from the maximum distance of the feature points to the initial surface (4-5 voxels). For the reconstructions in Figure 2, the mean and median distances (measured in voxel width) of the feature points to the reconstructed surface have been computed. See Figure 3 for a typical histogram and a table of the results. Most points have sub-voxel distance to the surface, i.e. the deviation is of the same order as the surface resolution. This shows that *we are not fitting a surface to the mean shape but that it really fits the feature data*. The reader is encouraged to zoom in on the reconstructions in Figure 2 to verify the quality.

## 5 Conclusions and Future Work

In this paper, a framework for reconstructing surface geometry based on learned shape models and level sets has been presented. The approach relies solely on images as input and the output of the system is a geometric 3D surface consistent with the input images. A new regularization approach is introduced in the level set framework, where the distance to an a priori 3D model is used instead of the common mean curvature regularization.

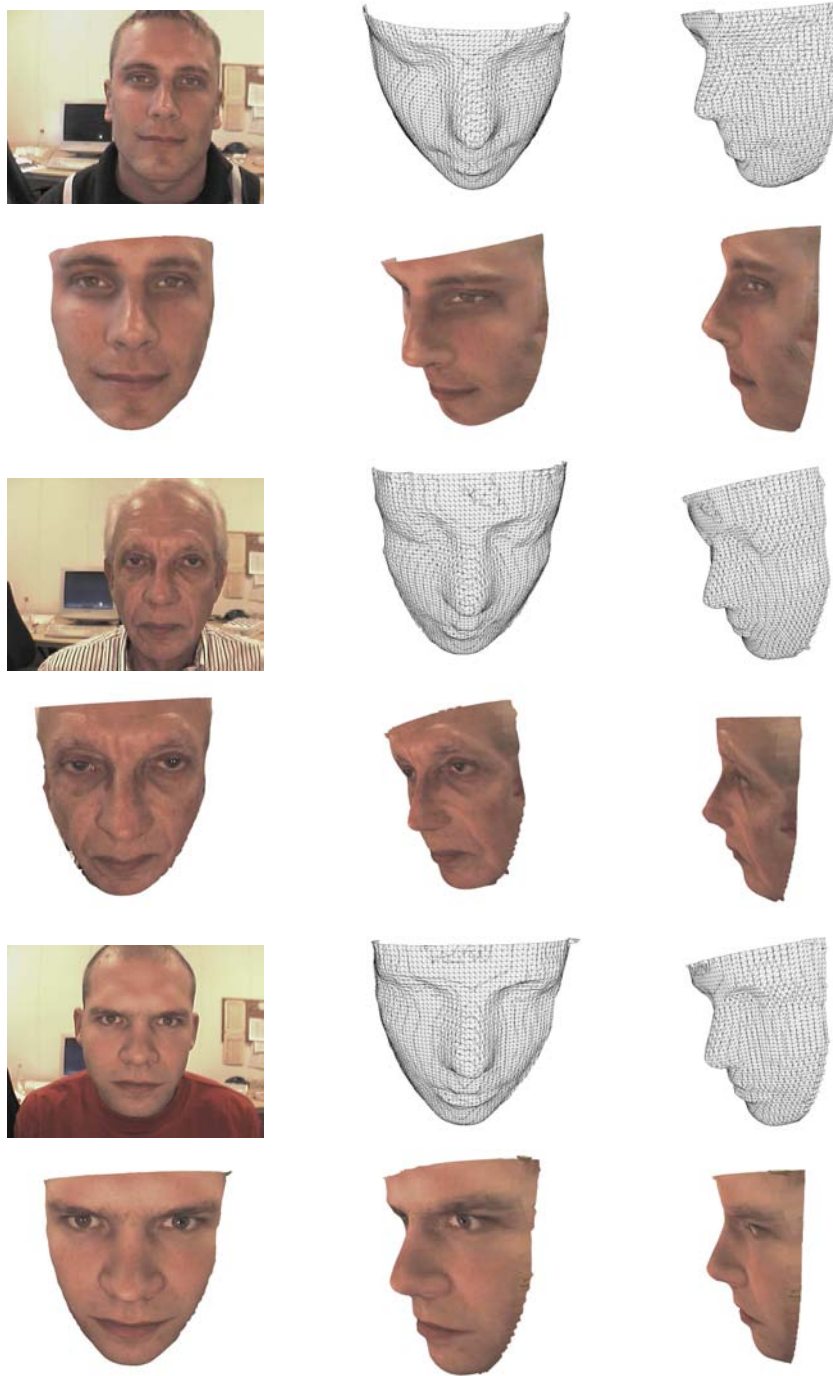

Figure 2: Input images and the reconstructed surface for three persons, two in the training data and one (bottom) in the test data. For each person: One of the input images, triangulated surfaces and surfaces with texture. Note that the profiles above are different from the ones in the input.

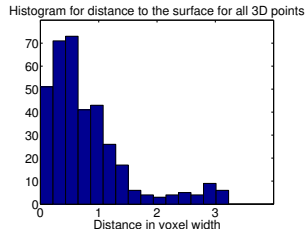

| | mean | median |
|---|---|---|
| Person 1 | 0.978 | 0.788 |
| Person 2 | 0.805 | 0.601 |
| Person 3 | 0.861 | 0.843 |

Figure 3: Histogram of the deviations of the feature points from the surface for the second person in Figure 2. The table displays mean and median deviations for all three persons.

Our current work focuses on incorporating a *robust* photo-consistency measure in the energy potential (7) to enable more detailed geometry. Also, the total number of faces is quite small in the database and we will collect and label more images. Currently, only images taken with the tri-stereo setup have been used with heads facing the middle camera (cf. Figure 1). Once the statistical model has been learned, it can be utilized for other (nearby) poses as well, but to what extent is yet to be explored.

## Footnotes

[1] In the experiments, $f(\cdot)$ will model the projection of three calibrated perspective cameras.

## References

[1] J.A. Sethian. *Level Set Methods and Fast Marching Methods Evolving Interfaces in Computational Geometry, Fluid Mechanics, Computer Vision, and Materials Science*. Cambridge University Press, 1999.

[2] T. F. Cootes and Taylor C. J. Active shape model search using local grey-level models: A quantatitative evaluation. In *British Machine Vision Conf.*, pages 639–648, 1993.

[3] T.F. Cootes, G.V. Wheeler, K.N. Walker, and C.J. Taylor. View-based active appearance models. *Image and Vision Computing*, 20(9-10):657–664, 2002.

[4] M. E. Tipping and C. M. Bishop. Probabilistic principal component analysis. *Phil. Trans. Royal Soc. London B*, 61(3):611–622, 1999.

[5] V. Blanz and T. Vetter. A morphable model for the synthesis of 3d faces. In *SIGGRAPH*, pages 187–194, 1999.

[6] S. Romdhani and T. Vetter. Efficient, robust and accurate fitting of a 3d morphable model. In *Int. Conf. Computer Vision*, pages 59–66, Nice, France, 2003.

[7] P. Fua. Regularized bundle-adjustment to model heads from image sequences without calibration data. *Int. J. Comput. Vision*, 38(2):153–171, 2000.

[8] B. Moghaddam, J. Lee, H. Pfister, and R. Machiraju. Model-based 3d face capture with shape-from-silhouettes. In *IEEE International Workshop on Analysis and Modeling of Faces and Gestures (AMFG)*, pages 20–27, 2003.

[9] H.K. Zhao, S. Osher, B. Merriman, and M. Kang. Implicit and non-parametric shape reconstruction from unorganized points using a variational level set method. In *Computer Vision and Image Understanding*, pages 295–319, 2000.

[10] J.E. Solem and F. Kahl. Surface reconstruction from the projection of points, curves and contours. In *2nd Int. Symposium on 3D Data Processing, Visualization and Transmission, Thessaloniki, Greece*, 2004.

[11] O. Faugeras and R. Keriven. Variational principles, surface evolution, PDEs, level set methods, and the stereo problem. *IEEE Transactions on Image Processing*, 7(3):336–344, 1998.

[12] M. Rousson and N. Paragios. Shape priors for level set representations. In *Proc. European Conf. on Computer Vision*, volume 2351 of *Lecture Notes in Computer Science*. Springer, 2002.

[13] K. Skoglund. Three-dimensional face modelling and analysis. Master's thesis, Informatics and Mathematical Modelling, Technical University of Denmark, DTU, 2003.

[14] J.E. Solem and A. Heyden. Reconstructing open surfaces from unorganized data points. In *International Conference on Computer Vision and Pattern Recognition, Washington DC*, 2004.
